# Matching Free Trees with Replicator Equations

**Marcello Pelillo**
Dipartimento di Informatica
Università Ca' Foscari di Venezia
Via Torino 155, 30172 Venezia Mestre, Italy
E-mail: `pelillo@dsi.unive.it`

## Abstract

Motivated by our recent work on rooted tree matching, in this paper we provide a solution to the problem of matching two free (i.e., unrooted) trees by constructing an association graph whose maximal cliques are in one-to-one correspondence with maximal common subtrees. We then solve the problem using simple replicator dynamics from evolutionary game theory. Experiments on hundreds of uniformly random trees are presented. The results are impressive: despite the inherent inability of these simple dynamics to escape from local optima, they always returned a globally optimal solution.

## 1  Introduction

Graph matching is a classic problem in computer vision and pattern recognition, instances of which arise in areas as diverse as object recognition, motion and stereo analysis [1]. In many problems (e.g., [2, 11, 19]) the graphs at hand have a peculiar structure: they are connected and acyclic, i.e. they are *free trees*. Note that, unlike "rooted" trees, in free trees there is no distinguished node playing the role of the root, and hence no hierarchy is imposed on them. Standard graph matching techniques, such as [8], yield solutions that are not constrained to preserve connectedness and hence cannot be applied to free trees.

A classic approach to solving the graph matching problem consists of transforming it into the equivalent problem of finding a maximum clique in an auxiliary graph structure, known as the *association graph* [1]. This framework is attractive because it casts graph matching as a pure graph-theoretic problem, for which a solid theory and powerful algorithms have been developed. Note that, although the maximum clique problem is known to be $NP$-hard, powerful heuristics exist which efficiently find good approximate solutions [4].

Motivated by our recent work on rooted tree matching [15], in this paper we propose a solution to the free tree matching problem by providing a straightforward way of deriving an association graph from two free trees. We prove that in the new formulation there is a one-to-one correspondence between maximal (maximum) cliques in the derived association graph and maximal (maximum) subtree isomorphisms. As an obvious corollary, the computational complexity of finding a maximum clique in such graphs is therefore the same as the subtree isomorphism problem, which is known to be polynomial in the number of nodes [7].

Following [13, 15], we use a recent generalization of the Motzkin-Straus theorem [12] to

formulate the maximum clique problem as a quadratic programming problem. To (approximately) solve it we employ *replicator equations*, a class of simple continuous- and discrete-time dynamical systems developed and studied in evolutionary game theory [10, 17].

We illustrate the power of the approach via experiments on hundreds of (uniformly) random trees. The results are impressive: despite the counter-intuitive maximum clique formulation of the tree matching problem, and the inherent inability of these simple dynamics to escape from local optima, they *always* found a globally optimal solution.

## 2 Subtree isomorphisms and maximal cliques

Let $G = (V, E)$ be a graph, where $V$ is the set of nodes and $E$ is the set of (undirected) edges. The *order* of $G$ is the number of nodes in $V$, while its *size* is the number of edges. Two nodes $u, v \in V$ are said to be *adjacent* (denoted $u \sim v$) if they are connected by an edge. The *adjacency matrix* of $G$ is the $n \times n$ symmetric matrix $A_G = (a_{ij})$ defined as

$$a_{ij} = \begin{cases} 1, & \text{if } v_i \sim v_j \\ 0, & \text{otherwise}. \end{cases}$$

The *degree* of a node $u$, denoted $\deg(u)$, is the number of nodes adjacent to it. A *path* is any sequence of distinct nodes $u_0 u_1 \ldots u_n$ such that for all $i = 1 \ldots n$, $u_{i-1} \sim u_i$; in this case, the *length* of the path is $n$. If $u_0 \sim u_n$ the path is called a *cycle*. A graph is said to be *connected* if any two nodes are joined by a path. The *distance* between two nodes $u$ and $v$, denoted by $d(u, v)$, is the length of the shortest path joining them (by convention $d(u, v) = \infty$, if there is no such path). Given a subset of nodes $C \subseteq V$, the *induced subgraph* $G[C]$ is the graph having $C$ as its node set, and two nodes are adjacent in $G[C]$ if and only if they are adjacent in $G$. A connected graph with no cycles is called a *free tree*, or simply a *tree*. Trees have a number of interesting properties. One which turns out to be very useful for our characterization is that in a tree any two nodes are connected by a *unique* path.

Let $T_1 = (V_1, E_1)$ and $T_2 = (V_2, E_2)$ be two trees. Any bijection $\phi : H_1 \rightarrow H_2$, with $H_1 \subseteq V_1$ and $H_2 \subseteq V_2$, is called a *subtree isomorphism* if it preserves both the adjacency relationships between the nodes and the connectedness of the matched subgraphs. Formally, this means that, given $u, v \in H_1$, we have $u \sim v$ if and only if $\phi(u) \sim \phi(v)$ and, in addition, the induced subgraphs $T_1[H_1]$ and $T_2[H_2]$ are connected. A subtree isomorphism is *maximal* if there is no other subtree isomorphism $\phi' : H'_1 \rightarrow H'_2$ with $H_1$ a strict subset of $H'_1$, and *maximum* if $H_1$ has largest cardinality. The maximal (maximum) subtree isomorphism problem is to find a maximal (maximum) subtree isomorphism between two trees. A word of caution about terminology is in order here. Despite name similarity, we are not addressing the so-called subtree isomorphism problem, which consists of determining whether a given tree is isomorphic to a subtree of a larger one. In fact, we are dealing with a generalization thereof, the maximum common subtree problem, which consists of determining the largest isomorphic subtrees of two given trees. We shall continue to use our own terminology, however, as it emphasizes the role of the isomorphism $\phi$.

The *free tree association graph* (FTAG) of two trees $T_1 = (V_1, E_1)$ and $T_2 = (V_2, E_2)$ is the graph $G = (V, E)$ where

$$V = V_1 \times V_2 \tag{1}$$

and, for any two nodes $(u, w)$ and $(v, z)$ in $V$, we have

$$(u, w) \sim (v, z) \Leftrightarrow d(u, v) = d(w, z) . \tag{2}$$

Note that this definition of the association graph is stronger than the standard one used for matching arbitrary relational structures [1].

A subset of vertices of $G$ is said to be a *clique* if all its nodes are mutually adjacent. A *maximal* clique is one which is not contained in any larger clique, while a *maximum* clique is a clique having largest cardinality. The maximum clique problem is to find a maximum clique of $G$.

The following theorem, which is the basis of the work reported here, establishes a one-to-one correspondence between the maximum subtree isomorphism problem and the maximum clique problem.

**Theorem 1** *Any maximal (maximum) subtree isomorphism between two trees induces a maximal (maximum) clique in the corresponding FTAG, and vice versa.*

*Proof (outline).* Let $\phi : H_1 \to H_2$ be a maximal subtree isomorphism between trees $T_1$ and $T_2$, and let $G = (V, E)$ denote the corresponding FTAG. Let $C_\phi \subseteq V$ be defined as $C_\phi = \{(u, \phi(u)) : u \in H_1\}$. From the definition of a subtree isomorphism it follows that $\phi$ maps the path between any two nodes $u, v \in H_1$ onto the path joining $\phi(u)$ and $\phi(v)$. This clearly implies that $d(u, v) = d(\phi(u), \phi(v))$ for all $u \in H_1$, and therefore $C_\phi$ is a clique. Trivially, $C_\phi$ is a maximal clique because $\phi$ is maximal, and this proves the first part of the theorem.

Suppose now that $C = \{(u_1, w_1), \cdots, (u_n, w_n)\}$ is a maximal clique of $G$, and let $H_1 = \{u_1, \cdots, u_n\} \subseteq V_1$ and $H_2 = \{w_1, \cdots, w_n\} \subseteq V_2$. Define $\phi : H_1 \to H_2$ as $\phi(u_i) = w_i$, for all $i = 1 \ldots n$. From the definition of the FTAG and the hypothesis that $C$ is a clique, it is simple to see that $\phi$ is a one-to-one and onto correspondence between $H_1$ and $H_2$, which trivially preserves the adjacency relationships between nodes. The fact that $\phi$ is a maximal isomorphism is a straightforward consequence of the maximality of $C$.

To conclude the proof we have to show that the subgraphs that we obtain when we restrict ourselves to $H_1$ and $H_2$, i.e. $T_1[H_1]$ and $T_2[H_2]$, are trees, and this is equivalent to showing that they are connected. Suppose by contradiction that this is not the case, and let $u_i, u_j \in H_1$ be two nodes which are not joined by a path in $T_1[H_1]$. Since both $u_i$ and $u_j$ are nodes of $T_1$, however, there must exist a path $u_i = x_0 x_1 \ldots x_m = u_j$ joining them in $T_1$. Let $x^* = x_k$, for some $k = 1 \ldots m$, be a node on this path which is not in $H_1$. Moreover, let $y^* = y_k$ be the $k$-th node on the path $w_i = y_0 y_1 \ldots y_m = w_j$ which joins $w_i$ and $w_j$ in $T_2$ (remember that $d(u_i, u_j) = d(w_i, w_j)$, and hence $d(w_i, w_j) = m$). It is easy to show that the set $\{(x^*, y^*)\} \cup C \subseteq V$ is a clique, thereby contradicting the hypothesis that $C$ is a maximal clique. This can be proved by exploiting the obvious fact that if $x$ is a node on the path joining any two nodes $u$ and $v$, then $d(u, v) = d(u, x) + d(x, v)$.

The "maximum" part of the statement is proved similarly. $\qquad\square$

The FTAG is readily derived by using a classical representation for graphs, i.e., the so-called *distance matrix* which, for an arbitrary graph $G = (V, E)$ of order $n$, is the $n \times n$ matrix $D = (d_{ij})$ where $d_{ij} = d(u_i, u_j)$, the distance between nodes $u_i$ and $u_j$. Efficient, classical algorithms are available for obtaining such a matrix [6]. Note also that the distance matrix of a graph can easily be constructed from its adjacency matrix $A_G$. In fact, denoting by $a_{ij}^n$ the $(i, j)$-th entry of the matrix $A_G^n$, the $n$-th power of $A_G$, we have that $d_{ij}$ equals the least $n$ for which $a_{ij}^n > 0$ (there must be such an $n$ since a tree is connected).

## 3    Matching free trees with replicator dynamics

Let $G = (V, E)$ be an arbitrary graph of order $n$, and let $S_n$ denote the standard simplex of $\mathbb{R}^n$:

$$S_n = \{ \, \mathbf{x} \in \mathbb{R}^n \; : \; \mathbf{e}'\mathbf{x} = 1 \text{ and } x_i \geq 0, \; i = 1 \ldots n \, \}$$

where **e** is the vector whose components equal 1, and a prime denotes transposition. Given a subset of vertices $C$ of $G$, we will denote by $\mathbf{x}^c$ its *characteristic vector* which is the point in $S_n$ defined as

$$x_i^c = \begin{cases} 1/|C|, & \text{if } i \in C \\ 0, & \text{otherwise} \end{cases}$$

where $|C|$ denotes the cardinality of $C$.

Now, consider the following quadratic function

$$f_G(\mathbf{x}) = \mathbf{x}' A_G \mathbf{x} + \frac{1}{2} \mathbf{x}' \mathbf{x} \tag{3}$$

where $A_G = (a_{ij})$ is the adjacency matrix of $G$. The following theorem, recently proved by Bomze [3], expands on the Motzkin-Straus theorem [12], a remarkable result which establishes a connection between the maximum clique problem and quadratic programming.

**Theorem 2** *Let $C$ be a subset of vertices of a graph $G$, and let $\mathbf{x}^c$ be its characteristic vector. Then, $C$ is a maximal (maximum) clique of $G$ if and only if $\mathbf{x}^c$ is a local (global) maximizer $f_G$ in $S_n$. Moreover, all local (and hence global) maximizers of $f_G$ in $S_n$ are strict and are characteristic vectors of maximal cliques of $G$.*

Unlike the original Motzkin-Straus formulation, which is plagued by the presence of "spurious" solutions [14], the previous result guarantees us that *all* maximizers of $f_G$ on $S_n$ are strict, and are characteristic vectors of maximal/maximum cliques in $G$. In a formal sense, therefore, a one-to-one correspondence exists between maximal cliques and local maximizers of $f_G$ in $S_n$ on the one hand, and maximum cliques and global maximizers on the other hand.

We now turn our attention to a class of simple dynamical systems that we use for solving our quadratic optimization problem. Let $W = (w_{ij})$ be a non-negative real-valued $n \times n$ matrix, and consider the following continuous-time dynamical system:

$$\dot{x}_i(t) = x_i(t) \left( \pi_i(t) - \sum_{j=1}^n x_j(t) \pi_j(t) \right) \tag{4}$$

where a dot signifies derivative with respect to time, and its discrete-time counterpart:

$$x_i(t+1) = \frac{x_i(t) \pi_i(t)}{\sum_{j=1}^n x_j(t) \pi_j(t)} \tag{5}$$

where

$$\pi_i(t) = \sum_{j=1}^n w_{ij} x_j(t) \ . \tag{6}$$

Both (4) and (5) are called *replicator equations* in evolutionary game theory, since they are used to model evolution over time of relative frequencies of interacting, self-replicating entities [10, 17]. It is readily seen that the simplex $S_n$ is invariant under these dynamics, which means that every trajectory starting in $S_n$ will remain in $S_n$ for all future times, and their stationary points coincide.

We are now interested in the dynamical properties of replicator dynamics; it is these properties that will allow us to solve our original tree matching problem. The following result is known in mathematical biology as the fundamental theorem of natural selection [10, 17] and, in its original form, traces back to R. A. Fisher.

**Theorem 3** *If* $W = W'$ *then the function* $\mathbf{x}'W\mathbf{x}$ *is strictly increasing along any non-constant trajectory under both continuous-time (4) and discrete-time (5) replicator dynamics. Furthermore, any such trajectory converges to a stationary point. Finally, a vector* $\mathbf{x} \in S_n$ *is asymptotically stable under (4) and (5) if and only if* $\mathbf{x}$ *is a strict local maximizer of* $\mathbf{x}'W\mathbf{x}$ *on* $S_n$.

In light of their dynamical properties, replicator equations naturally suggest themselves as a simple heuristic for solving the maximal subtree isomorphism problem. Indeed, let $T_1 = (V_1, E_1)$ and $T_2 = (V_2, E_2)$ be two free trees, and let $A_G$ denote the adjacency matrix of their FTAG $G$. By letting

$$W = A_G + \frac{1}{2}I \tag{7}$$

where $I$ is the identity matrix, we know that the replicator dynamical systems (4) and (5), starting from an arbitrary initial state, will iteratively maximize the function $f_G$ defined in (3) over the simplex and will eventually converge with probability 1 to a strict local maximizer which, by virtue of Theorem 2, will then correspond to the characteristic vector of a maximal clique in the association graph. As stated in Theorem 1, this will in turn induce a maximal subtree isomorphism between $T_1$ and $T_2$. Clearly, in theory there is no guarantee that the converged solution will be a *global* maximizer of $f_G$, and therefore that it will induce a *maximum* isomorphism between the two original trees, but see below.

Recently, there has been much interest around the following exponential version of replicator equations, which arises as a model of evolution guided by imitation [9, 10, 17]:

$$\dot{x}_i(t) = x_i(t) \left( \frac{e^{\kappa \pi_i(t)}}{\sum_{j=1}^{n} x_j(t) e^{\kappa \pi_j(t)}} - 1 \right) , \tag{8}$$

where $\kappa$ is a positive constant. As $\kappa$ tends to 0, the orbits of this dynamics approach those of the standard, "first-order" replicator model (4), slowed down by the factor $\kappa$; moreover, for large values of $\kappa$ the model approximates the so-called "best-reply" dynamics [9, 10]. A customary way of discretizing equation (8) is given by the following difference equations:

$$x_i(t+1) = \frac{x_i(t) e^{\kappa \pi_i(t)}}{\sum_{j=1}^{n} x_j(t) e^{\kappa \pi_j(t)}} . \tag{9}$$

From a computational perspective, exponential replicator dynamics are particularly attractive as they may be considerably faster and even more accurate than the standard, first-order model (see [13] and the experiments reported in the next section).

## 4   Results and conclusions

We tested our algorithms over large random trees. Random structures represent a useful benchmark not only because they are not constrained to any particular application, but also because it is simple to replicate experiments and hence to make comparisons with other algorithms.

In this series of experiments, the following protocol was used. A hundred 100-node free trees were generated uniformly at random using a procedure described by Wilf in [18]. Then, each such tree was subject to a corruption process which consisted of randomly deleting a fraction of its nodes (in fact, the to-be-deleted nodes were constrained to be the terminal ones, otherwise the resulting graph would have been disconnected), thereby obtaining a tree isomorphic to a proper subtree of the original one. Various levels of corruption (i.e., percentage of node deletion) were used, namely 2%, 10%, 20%, 30% and 40%. This means that the order of the pruned trees ranged from 98 to 60. Overall, therefore, 500

pairs of trees were obtained, for each of which the corresponding FTAG was constructed as described in Section 2. To keep the order of the association graph as low as possible, its vertex set was constructed as follows:

$$V = \{(u, w) \in V' \times V'' \ : \ \deg(u) \leq \deg(w)\} \ ,$$

assuming $|V'| \leq |V''|$, the edge set $E$ being defined as in (2). It is straightforward to see that when the first tree is isomorphic to a subtree of the second, Theorem 1 continues to hold. This simple heuristic may significantly reduce the dimensionality of the search space. We also performed some experiments with unpruned FTAG's but no significant difference in performance was noticed apart, of course, heavier memory requirements.

Both the discrete-time first-order dynamics (5) and its exponential counterpart (9) (with $\kappa = 10$) were used. The algorithms were started from the simplex barycenter and stopped when either a maximal clique (i.e., a local maximizer of $f_G$) was found or the distance between two successive points was smaller than a fixed threshold. In the latter case the converged vector was randomly perturbed, and the algorithms restarted from the perturbed point. Note that this situation corresponds to convergence to a saddle point.

After convergence, we calculated the proportion of matched nodes, i.e., the ratio between the cardinality of the clique found and the order of the smaller subtree, and then we averaged. Figure 1(a) shows the results obtained using the linear dynamics (5) as a function of the corruption level. As can be seen, the algorithm was *always* able to find a correct maximum isomorphism, i.e. a maximum clique in the FTAG. Figure 1(b) plots the corresponding (average) CPU time taken by the processes, with corresponding error bars (simulations were performed on a machine equipped with a 350MHz AMDK6-2 processor).

In Figure 2, the results pertaining to the exponential dynamics (8) are shown. In terms of solution's quality the algorithm performed exactly as its linear counterpart, but this time it was dramatically faster. This confirms earlier results reported in [13].

Before concluding, we note that our approach can easily be extended to tackle the problem of matching attributed (free) trees. In this case, the attributes result in weights being placed on the nodes of the association graph, and a conversion of the maximum clique problem to a maximum weight clique problem [15, 5]. Moreover, it is straightforward to formulate error-tolerant versions of our framework along the lines suggested in [16] for rooted attributed trees, where many-to-many node correspondences are allowed. All this will be the subject of future investigations.

Finally, we think that the results presented in this paper (together with those reported in [13, 15]) raise intriguing questions concerning the connections between (standard) notions of computational complexity and the "elusiveness" of global optima in continuous settings.

**Acknowledgments.** The author would like to thank M. Zuin for his support in performing the experiments.

# References

[1] D. H. Ballard and C. M. Brown. *Computer Vision.* Prentice-Hall, Englewood Cliffs, NJ, 1982.

[2] H. Blum and R. N. Nagel. Shape description using weighted symmetric axis features. *Pattern Recognition*, 10:167–180, 1978.

[3] I. M. Bomze. Evolution towards the maximum clique. *J. Glob. Optim.*, 10:143–164, 1997.

[4] I. M. Bomze, M. Budinich, P. M. Pardalos, and M. Pelillo. The maximum clique problem. In D.-Z. Du and P. M. Pardalos, editors, *Handbook of Combinatorial Optimization (Suppl. Vol. A)*, pages 1–74. Kluwer, Boston, MA, 1999.

[5] I. M. Bomze, M. Pelillo, and V. Stix. Approximating the maximum weight clique using replicator dynamics. *IEEE Trans. Neural Networks*, 11(6):1228–1241, 2000.

Figure 1: Results obtained over 100-node random trees with various levels of corruption, using the first-order dynamics (5). Top: Percentage of correct matches. Bottom: Average computational time taken by the replicator equations.

[6] T. H. Cormen, C. E. Leiserson, and R. L. Rivest. *Introduction to Algorithms*. MIT Press, Cambridge, MA, 1990.

[7] M. R. Garey and D. S. Johnson. *Computers and Intractability: A Guide to the Theory of NP-Completeness*. W. H. Freeman, San Francisco, CA, 1979.

[8] S. Gold and A. Rangarajan. A graduated assignment algorithm for graph matching. *IEEE Trans. Pattern Anal. Machine Intell.* 18:377-388, 1996.

[9] J. Hofbauer. Imitation dynamics for games. Collegium Budapest, preprint, 1995.

[10] J. Hofbauer and K. Sigmund. *Evolutionary Games and Population Dynamics*. Cambridge University Press, Cambridge, UK, 1998.

[11] T.-L. Liu, D. Geiger, and R. V. Kohn. Representation and self-similarity of shapes. In *Proc. ICCV'98—6th Int. Conf. Computer Vision*, pages 1129–1135, Bombay, India, 1998.

[12] T. S. Motzkin and E. G. Straus. Maxima for graphs and a new proof of a theorem of Turán. *Canad. J. Math.*, 17:533–540, 1965.

Figure 2: Results obtained over 100-node random trees with various levels of corruption, using the exponential dynamics (9) with $\kappa = 10$. Top: Percentage of correct matches. Bottom: Average computational time taken by the replicator equations.

[13] M. Pelillo. Replicator equations, maximal cliques, and graph isomorphism. *Neural Computation*, 11(8):2023–2045, 1999.

[14] M. Pelillo and A. Jagota. Feasible and infeasible maxima in a quadratic program for maximum clique. *J. Artif. Neural Networks*, 2:411–420, 1995.

[15] M. Pelillo, K. Siddiqi, and S. W. Zucker. Matching hierarchical structures using association graphs. *IEEE Trans. Pattern Anal. Machince Intell.*, 21(11):1105–1120, 1999.

[16] M. Pelillo, K. Siddiqi, and S. W. Zucker. Many-to-many matching of attributed trees using association graphs and game dynamics. In C. Arcelli, L. P. Cordella, and G. Sanniti di Baja, editors, *Visual Form 2001*, pages 583–593. Springer, Berlin, 2001.

[17] J. W. Weibull. *Evolutionary Game Theory*. MIT Press, Cambridge, MA, 1995.

[18] H. Wilf. The uniform selection of free trees. *J. Algorithms*, 2:204–207, 1981.

[19] S. C. Zhu and A. L. Yuille. FORMS: A flexible object recognition and modeling system. *Int. J. Computer Vision*, 20(3):187–212, 1996.
